# Boosting with Maximum Adaptive Sampling

**Charles Dubout**
Idiap Research Institute
charles.dubout@idiap.ch

**François Fleuret**
Idiap Research Institute
francois.fleuret@idiap.ch

## Abstract

Classical Boosting algorithms, such as AdaBoost, build a strong classifier without concern about the computational cost. Some applications, in particular in computer vision, may involve up to millions of training examples and features. In such contexts, the training time may become prohibitive. Several methods exist to accelerate training, typically either by sampling the features, or the examples, used to train the weak learners. Even if those methods can precisely quantify the speed improvement they deliver, they offer no guarantee of being more efficient than any other, given the same amount of time.

This paper aims at shading some light on this problem, i.e. given a fixed amount of time, for a particular problem, which strategy is optimal in order to reduce the training loss the most. We apply this analysis to the design of new algorithms which estimate on the fly at every iteration the optimal trade-off between the number of samples and the number of features to look at in order to maximize the expected loss reduction. Experiments in object recognition with two standard computer vision data-sets show that the adaptive methods we propose outperform basic sampling and state-of-the-art bandit methods.

## 1   Introduction

Boosting is a simple and efficient machine learning algorithm which provides state-of-the-art performance on many tasks. It consists of building a strong classifier as a linear combination of weak-learners, by adding them one after another in a greedy manner. However, while textbook AdaBoost repeatedly selects each of them using all the training examples and all the features for a predetermined number of rounds, one is not obligated to do so and can instead choose only to look at a subset of examples and features.

For the sake of simplicity, we identify the space of weak learners and the feature space by considering all the thresholded versions of the latter. More sophisticated combinations of features can be envisioned in our framework by expanding the feature space.

The computational cost of one iteration of Boosting is roughly proportional to the product of the number of candidate weak learners $Q$ and the number of samples $T$ considered, and the performance increases with both. More samples allow a more accurate estimation of the weak-learners' performance, and more candidate weak-learners increase the performance of the best one. Therefore, one wants at the same time to look at a large number of candidate weak-learners, in order to find a good one, but also needs to look at a large number of training examples, to get an accurate estimate of the weak-learner performances. As Boosting progresses, the candidate weak-learners tend to behave more and more similarly, as their performance degrades. While a small number of samples is initially sufficient to characterize the good weak-learners, it becomes more and more difficult, and the optimal values for a fixed product $Q\,T$ moves to larger $T$ and smaller $Q$.

We focus in this paper on giving a clear mathematical formulation of the behavior described above. Our main analytical results are Equations (13) and (17) in § 3. They give exact expressions of the

expected edge of the selected weak-learner – that is the immediate loss reduction it provides in the considered Boosting iteration – as a function of the number $T$ of samples and number $Q$ of weak-learners used in the optimization process. From this result we derive several algorithms described in § 4, and estimate their performance compared to standard and state-of-the-art baselines in § 5.

## 2 Related works

The most computationally intensive operation performed in Boosting is the optimization of the weak-learners. In the simplest version of the procedure, it requires to estimate for each candidate weak-learner a score dubbed "edge", which requires to loop through every training example. Reducing this computational cost is crucial to cope with high-dimensional feature spaces or very large training sets. This can be achieved through two main strategies: sampling the training examples, or the feature space, since there is a direct relation between features and weak-learners.

Sampling the training set was introduced historically to deal with weak-learners which can not be trained with weighted samples. This procedure consists of sampling examples from the training set according to their Boosting weights, and of approximating a weighted average over the full set by a non-weighted average over the sampled subset. See § 3.1 for formal details. Such a procedure has been re-introduced recently for computational reasons [5, 8, 7], since the number of subsampled examples controls the trade-off between statistical accuracy and computational cost.

Sampling the feature space is the central idea behind LazyBoost [6], and consists simply of replacing the brute-force exhaustive search over the full feature set by an optimization over a subset produced by sampling uniformly a predefined number of features. The natural redundancy of most of the families of features makes such a procedure particularly efficient.

Recently developed methods rely on multi-arms bandit methods to balance properly the exploitation of features known to be informative, and the exploration of new features [3, 4]. The idea behind those methods is to associate a bandit arm to every feature, and to see the loss reduction as a reward. Maximizing the overall reduction is achieved with a standard bandit strategy such as **UCB** [1], or **Exp3.P** [2], see § 5.2 for details.

These techniques suffer from three important drawbacks. First they make the assumption that the quality of a feature – the expected loss reduction of a weak-learner using it – is stationary. This goes against the underpinning of Boosting, which is that at any iteration the performance of the learners is relative to the sample weights, which evolves over the training (**Exp3.P** does not make such an assumption explicitly, but still rely only on the history of past rewards). Second, without additional knowledge about the feature space, the only structure they can exploit is the stationarity of individual features. Hence, improvement over random selection can only be achieved by sampling again *the exact same features* one has already seen in the past. We therefore only use those methods in a context where features come from multiple families. This allows us to model the quality, and to bias the sampling, at the level of families instead of individual features.

Those approaches exploit information about features to bias the sampling, hence making it more efficient, and reducing the number of weak-learners required to achieve the same loss reduction. However, they do not explicitly aim at controlling the computational cost. In particular, there is no notion of varying the number of samples used for the estimation of the weak-learners' performance.

## 3 Boosting with noisy maximization

We present in this section some analytical results to approximate a standard round of AdaBoost – or most other Boosting algorithms – by sampling both the training examples and the features used to build the weak-learners. Our main goal is to devise a way of selecting the optimal numbers of weak-learners $Q$ and samples $T$ to look at, so that their product is upper-bounded by a given constant, and that the expectation of the *real* performance of the selected weak-learner is maximal.

In § 3.1 we recall standard notation for Boosting, the concept of the edge of a weak-learner, and how it can be approximated by a sampling of the training examples. In § 3.2 we formalize the optimization of the learners and derive the expectation $E[G^*]$ of the true edge $G^*$ of the selected weak-learner, and we illustrate these results in the Gaussian case in § 3.3.

$\mathbf{1}_{\{\text{condition}\}}$ is equal to $1$ if the condition is true, $0$ otherwise
$N$  number of training examples
$F$  number of weak-learners
$K$  number of families of weak-learners
$T$  number of examples subsampled from the full training set
$Q$  number of weak-learners sampled in the case of a single family of features
$Q_1, \ldots, Q_K$  number of weak-learners sampled from each one of the $K$ families
$(x_n, y_n) \in \mathcal{X} \times \{-1, 1\}$  training examples
$\omega_n \in \mathbb{R}$  weights of the $n$th training example in the considered Boosting iteration
$G_q$  true edge of the $q$th weak-learner
$G^*$  true edge of the selected weak-learner
$e(Q, T)$  value of $E[G^*]$, as a function of $Q$ and $T$
$e(Q_1, \ldots, Q_K, T)$  value of $E[G^*]$, in the case of $K$ families of features
$H_q$  approximated edge of the $q$th weak-learner, estimated from the subsampled $T$ examples
$\Delta_q$  estimation error in the approximated edge $H_q - G_q$

Table 1: Notations

As stated in the introduction, we will ignore the feature space itself, and only consider in the following sections the set of weak-learners built from it. Also, note that both the Boosting procedure and our methods are presented in the context of binary classification, but can be easily extended to a multi-class context using for example AdaBoost.MH, which we used in all our experiments.

## 3.1  Edge estimation with weighting-by-sampling

Given a training set

$$(x_n, y_n) \in \mathcal{X} \times \{-1, 1\}, \; n = 1, \ldots, N \tag{1}$$

and a set $\mathcal{H}$ of weak-learners, the standard Boosting procedure consists of building a strong classifier

$$f(x) = \sum_i \alpha_i \, h_i(x) \tag{2}$$

by choosing the terms $\alpha_i \in \mathbb{R}$ and $h_i \in \mathcal{H}$ in a greedy manner to minimize a loss estimated over the training samples.

At every iteration, choosing the optimal weak-learner boils down to finding the weak-learner with the largest edge, which is the derivative of the loss reduction w.r.t. the weak-learner weight. The higher this value, the more the loss can be reduced locally, and thus the better the weak-learner. The edge is a linear function of the responses of the weak-learner over the samples

$$G(h) = \sum_{n=1}^{N} y_n \omega_n h(x_n), \tag{3}$$

where the $\omega_n$s depend on the current responses of $f$ over the $x_n$s. We consider without loss of generality that they have been normalized such that $\sum_{n=1}^{N} \omega_n = 1$.

Given an arbitrary distribution $\eta$ over the sample indexes, with a non-zero mass over every index, we can rewrite the edge as

$$G(h) = E_{N \sim \eta} \left[ \frac{y_N \omega_N}{\eta(N)} \, h(x_N) \right] \tag{4}$$

which, for $\eta(n) = \omega_n$ gives

$$G(h) = E_{N \sim \eta} \left[ y_N h(x_N) \right] \tag{5}$$

The idea of weighting-by-sampling consists of replacing the expectation in that expression with an approximation obtained by sampling. Let $N_1, \ldots, N_T$, be i.i.d. of distribution $\eta$, we define the approximated edge as

$$H(h) = \frac{1}{T} \sum_{t=1}^{T} y_{N_t} h(x_{N_t}), \tag{6}$$

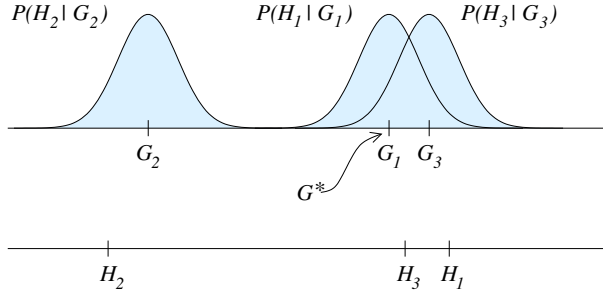

Figure 1: To each of the $Q$ weak-learner corresponds a real edge $G_q$ computed over all the training examples, and an approximated edge $H_q$ computed from a subsampling of $T$ training examples. The approximated edge fluctuates around the true value, with a binomial distribution. The Boosting algorithm selects the weak-learner with the highest approximated edge, which has a real edge $G^*$. On this figure, the largest approximated edge is $H_1$, hence the real edge $G^*$ of the selected weak-learner is equal to $G_1$, which is less than $G_3$.

which follows a binomial distribution centered on the true edge, with a variance decreasing with the number of samples $T$. It is accurately modeled with

$$H(h) \sim \mathcal{N}\left(G, \frac{(1+G)(1-G)}{T}\right). \tag{7}$$

## 3.2 Formalization of the noisy maximization

Let $G_1, \ldots, G_Q$ be a series of independent, real-valued random variables standing for the true edges of $Q$ weak-learners sampled randomly. Let $\Delta_1, \ldots, \Delta_Q$ be a series of independent, real-valued random variables standing for the noise in the estimation of the edges due to the sampling of only $T$ training examples, and finally $\forall q$, let $H_q = G_q + \Delta_q$ be the approximated edge.

We define $G^*$ as the true edge of the weak-learner, which has the highest approximated edge

$$G^* = G_{\mathrm{argmax}_{1 \le q \le Q} H_q} \tag{8}$$

This quantity is random due to both the sampling of the weak-learners, and the sampling of the training examples.

The quantity we want to optimize is $e(Q, T) = E[G^*]$, the expectation of the true edge of the selected learner, which increases with both $Q$ and $T$. A higher $Q$ increases the number of terms in the maximization of Equation (8), and a higher $T$ reduces the variance of the $\Delta$s, ensuring that $G^*$ is close to $\max_q G_q$. In practice, if the variance of the $\Delta$s is of the order of, or higher than, the variance of the $G$s, the maximization is close to a pure random selection, and looking at many weak-learners is useless.

We have:

$$e(Q, T) = E[G^*] \tag{9}$$

$$= E\left[G_{\mathrm{argmax}_{1 \le q \le Q} H_q}\right] \tag{10}$$

$$= \sum_{q=1}^{Q} E\left[G_q \prod_{u \ne q} \mathbf{1}_{\{H_q > H_u\}}\right] \tag{11}$$

$$= \sum_{q=1}^{Q} E\left[E\left[G_q \prod_{u \ne q} \mathbf{1}_{\{H_q > H_u\}} \,\middle|\, H_q\right]\right] \tag{12}$$

$$= \sum_{q=1}^{Q} E\left[E[G_q | H_q] \prod_{u \ne q} E\left[\mathbf{1}_{\{H_q > H_u\}} \,\middle|\, H_q\right]\right] \tag{13}$$

If the distributions of the $G_q$s and the $\Delta_q$s are Gaussians or mixtures of Gaussians, we can derive analytical expressions for both $E[G_q|H_q]$ and $E\big[\mathbf{1}_{\{H_q>H_u\}}\,\big|\,H_q\big]$, and compute the value of $e(Q,T)$ efficiently.

In the case of multiple families of weak-learners, it makes sense to model the distributions of the edges $G_q$ separately for each family, as they often have a more homogeneous behavior inside a family than across families. We can easily adapt the framework developed in the previous sections to that case, and we define $e(Q_1,\ldots,Q_K,T)$, the expected edge of the selected weak-learner when we sample $T$ examples from the training set, and $Q_k$ weak-learners from the $k$th family.

### 3.3 Gaussian case

As an illustrative example, we consider here the case where the $G_q$s, the $\Delta_q$s, and hence also the $H_q$s all follow Gaussian distributions. We take $G_q \sim \mathcal{N}(0,1)$, and $\Delta_q \sim \mathcal{N}(0,\sigma^2)$, and obtain:

$$e(Q,T) = Q\,E\left[ E[G_1|H_1] \prod_{u\neq 1} E\big[\mathbf{1}_{\{H_1>H_u\}}\,\big|\,H_1\big] \right] \tag{14}$$

$$= Q\,E\left[ \frac{H_1}{\sigma^2+1}\Phi\left(\frac{H_1}{\sqrt{\sigma^2+1}}\right)^{Q-1} \right] \tag{15}$$

$$= \frac{1}{\sqrt{\sigma^2+1}}\,E\Big[Q\,G_1\Phi\left(G_1\right)^{Q-1}\Big] \tag{16}$$

$$= \frac{1}{\sqrt{\sigma^2+1}}\,E\left[\max_{1\leq q\leq Q} G_q\right]. \tag{17}$$

Where, $\Phi$ stands for the cumulative distribution function of the unit Gaussian, and $\sigma$ depends on $T$. See Figure 2 for an illustration of the behavior of $e(Q,T)$ for two different variances of the $G_q$s.

There is no reason to expect the distribution of the $G_q$s to be Gaussian, contrary to the $\Delta_q$s, as shown by Equation (7), but this is not a problem as it can always be approximated by a mixture, for which we can still derive analytical expressions, even if the $G_q$s or the $\Delta_q$s have different distributions for different $q$s.

## 4 Adaptive Boosting Algorithms

We propose here several new algorithms to sample features and training examples adaptively at each Boosting step.

While all the formulation above deals with uniform sampling of weak learners, we actually sample features, and optimize thresholds to build stumps. We observed that after a small number of Boosting iterations, the Gaussian model of Equation (7) is sufficiently accurate.

### 4.1 Maximum Adaptive Sampling

At every Boosting step, our first algorithm **MAS Naive** models $G_q$ with a Gaussian mixture model fitted on the edges estimated at the previous iteration, computes from that density model the pair $(Q,T)$ maximizing $e(Q,T)$, samples the corresponding number of examples and features, and keeps the weak-learner with the highest approximated edge.

The algorithm **MAS 1.Q** takes into account the decomposition of the feature set into $K$ families of feature extractors. It models the distributions of the $G_q$s separately, estimating the distribution of each on a small number of features and examples sampled at the beginning of each iteration, chosen so as to account for 10% of the total cost. From these models, it optimizes $Q$, $T$ and the index $l$ of the family to sample from, to maximize $e(Q\mathbf{1}_{\{l=1\}},\ldots,Q\,\mathbf{1}_{\{l=K\}},T)$. Hence, in a given Boosting step, it does not mix weak-learners based on features from different families.

Finally **MAS Q.1** similarly models the distributions of the $G_q$s, but it optimizes $Q_1,\ldots,Q_K$, $T$ greedily, starting from $Q_1 = 0,\ldots,Q_K = 0$, and iteratively incrementing one of the $Q_l$ so as to maximize $e(Q_1,\ldots,Q_K,T)$.

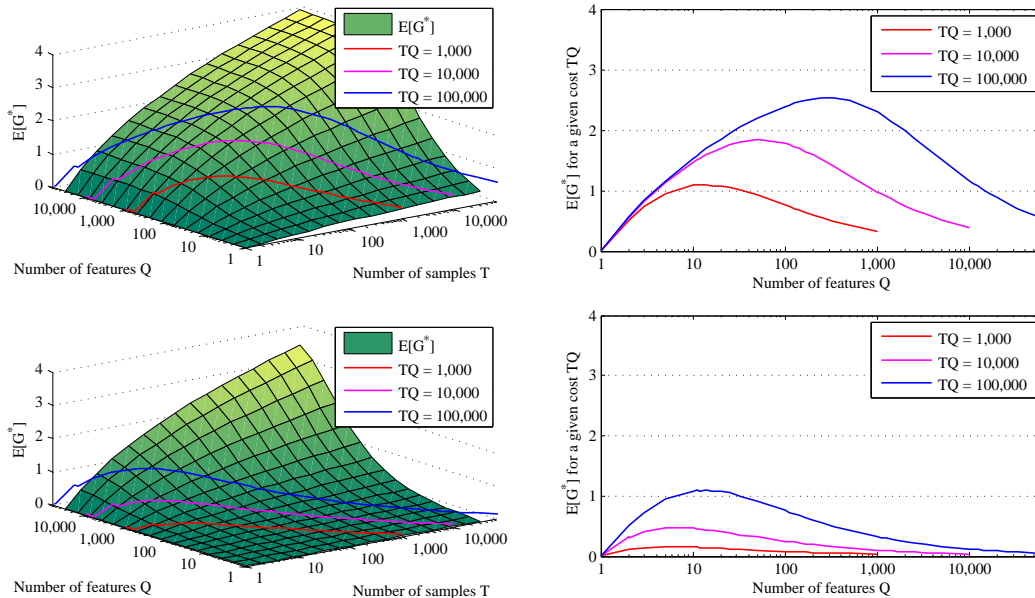

Figure 2: Simulation of the expectation of $G^*$ in the case where both the $G_q$s and the $\Delta_q$s follow Gaussian distributions. Top: $G_q \sim \mathcal{N}(0, 10^{-2})$. Bottom: $G_q \sim \mathcal{N}(0, 10^{-4})$. In both simulations $\Delta_q \sim \mathcal{N}(0, 1/T)$. Left: Expectation of $G^*$ vs. the number of sampled weak-learner $Q$ and the number of samples $T$. Right: same value as a function of $Q$ alone, for different fixed costs (product of the number of examples $T$ and $Q$). As this graphs illustrates, the optimal value for $Q$ is greater for larger variances of the $G_q$. In such a case the $G_q$ are more spread out, and identifying the largest one can be done despite a large noise in the estimations, hence with a limited number of samples.

### 4.2 Laminating

The fourth algorithm we have developed tries to reduce the requirement for a density model of the $G_q$. At every Boosting step it iteratively reduces the number of considered weak-learners, and increases the number of samples.

More precisely: given a fixed $Q_0$ and $T_0$, at every Boosting iteration, the **Laminating** first samples $Q_0$ weak-learners and $T_0$ training examples. Then, it computes the approximated edges and keeps the $Q_0/2$ best weak-learners. If more than one remains, it samples $2T_0$ examples, and re-iterates. The cost of each iteration is constant, equal to $Q_0 T_0$, and there are at most $\log_2(Q_0)$ of them, leading to an overall cost of $O(\log_2(Q_0)Q_0 T_0)$. In the experiments, we equalize the computational cost with the MAS approaches parametrized by $T, Q$ by forcing $\log_2(Q_0)Q_0 T_0 = TQ$.

## 5 Experiments

We demonstrate the validity of our approach for pattern recognition on two standard data-sets, using multiple types of image features. We compare our algorithms both to different flavors of uniform sampling and to state-of-the-art bandit based methods, all tuned to deal properly with multiple families of features.

### 5.1 Datasets and features

For the first set of experiments we use the well known MNIST handwritten digits database [10], containing respectively 60,000/10,000 train/test grayscale images of size $28 \times 28$ pixels, divided in ten classes. We use features computed by multiple image descriptors, leading to a total of 16,451 features. Those descriptors can be broadly divided in two categories. (1) **Image transforms**: Identity, Gradient image, Fourier and Haar transforms, Local Binary Patterns (LBP/iLBP). (2) **Histograms**:

sums of the intensities in random image patches, histograms of (oriented and non oriented) gradients at different locations, Haar-like features.

For the second set of experiments we use the challenging CIFAR-10 data-set [9], a subset of the 80 million tiny images data-set. It contains respectively 50,000/10,000 train/test color images of size $32 \times 32$ pixels, also divided in 10 classes. We dub it as challenging as state-of-the-art results without using additional training data barely reach 65% accuracy. We use directly as features the same image descriptors as described above for MNIST, plus additional versions of some of them making use of color information.

## 5.2  Baselines

We first define three baselines extending LazyBoost in the context of multiple feature families. The most naive strategy one could think of, that we call **Uniform Naive**, simply ignores the families, and picks features uniformly at random. This strategy does not properly distribute the sampling among the families, thus if one of them had a far greater cardinality than the others, all features would come from it. We define **Uniform 1.Q** to pick one of the feature families at random, and then samples the $Q$ features from that single family, and **Uniform Q.1** to pick uniformly at random $Q$ families of features, and then pick one feature uniformly in each family.

The second family of baselines we have tested bias their sampling at every Boosting iteration according to the observed edges in the previous iterations, and balance the exploitation of families of features known to perform well with the exploration of new family by using bandit algorithms [3, 4]. We use three such baselines (**UCB**, **Exp3.P**, $\epsilon$-**greedy**), which differ only by the underlying bandit algorithm used.

We tune the meta-parameters of these techniques – namely the scale of the reward and the exploration-exploitation trade-off – by training them multiple times over a large range of parameters and keeping only the results of the run with the smallest final Boosting loss. Hence, the computational cost is around one order of magnitude higher than for our methods in the experiments.

| Nb. of stumps | Uniform | | | Bandits | | | MAS | | | Laminating |
|---|---|---|---|---|---|---|---|---|---|---|
| | **Naive** | **1.Q** | **Q.1** | **UCB***| **Exp3.P***| $\epsilon$-**greedy***| **Naive** | **1.Q** | **Q.1** | |
| MNIST | | | | | | | | | | |
| 10 | -0.34 (0.01) | -0.33 (0.02) | -0.35 (0.02) | -0.33 (0.01) | -0.32 (0.01) | -0.34 (0.02) | -0.51 (0.02) | -0.50 (0.02) | -0.52 (0.01) | -0.43 (0.00) |
| 100 | -0.80 (0.01) | -0.73 (0.03) | -0.81 (0.01) | -0.73 (0.01) | -0.73 (0.02) | -0.73 (0.03) | -1.00 (0.01) | -1.00 (0.01) | -1.03 (0.01) | -1.01 (0.01) |
| 1,000 | -1.70 (0.01) | -1.45 (0.02) | -1.68 (0.01) | -1.64 (0.01) | -1.52 (0.02) | -1.60 (0.04) | -1.83 (0.01) | -1.80 (0.01) | -1.86 (0.00) | -1.99 (0.01) |
| 10,000 | -5.32 (0.01) | -3.80 (0.02) | -5.04 (0.01) | -5.26 (0.01) | -5.35 (0.04) | -5.38 (0.09) | -5.35 (0.01) | -5.05 (0.02) | -5.30 (0.00) | -6.14 (0.01) |
| CIFAR-10 | | | | | | | | | | |
| 10 | -0.26 (0.00) | -0.25 (0.01) | -0.26 (0.00) | -0.25 (0.01) | -0.25 (0.01) | -0.26 (0.00) | -0.28 (0.00) | -0.28 (0.00) | -0.28 (0.01) | -0.28 (0.00) |
| 100 | -0.33 (0.00) | -0.33 (0.01) | -0.34 (0.00) | -0.33 (0.00) | -0.33 (0.00) | -0.33 (0.00) | -0.35 (0.00) | -0.35 (0.00) | -0.37 (0.01) | -0.37 (0.00) |
| 1,000 | -0.47 (0.00) | -0.46 (0.00) | -0.48 (0.00) | -0.48 (0.00) | -0.47 (0.00) | -0.48 (0.00) | -0.48 (0.00) | -0.48 (0.00) | -0.49 (0.01) | -0.50 (0.00) |
| 10,000 | -0.93 (0.00) | -0.85 (0.00) | -0.91 (0.00) | -0.90 (0.00) | -0.91 (0.00) | -0.91 (0.00) | -0.93 (0.00) | -0.88 (0.00) | -0.89 (0.01) | -0.90 (0.00) |

Table 2: Mean and standard deviation of the Boosting loss ($\log_{10}$) on the two data-sets and for each method, estimated on ten randomized runs. Methods highlighted with a $\star$ require the tuning of meta-parameters which have been optimized by training fully multiple times.

## 5.3  Results and analysis

We report the results of the proposed algorithms against the baselines introduced in § 5.2 on the two data-sets of § 5.1 using the standard train/test cuts in tables 2 and 3. We ran each configuration ten times and report the mean and standard deviation of each. We set the maximum cost of all the algorithms to $10N$, setting $Q = 10$ and $T = N$ for the baselines, as this configuration leads to the best results after 10,000 Boosting rounds of AdaBoost.MH.

These results illustrate the efficiency of the proposed methods. For 10, 100 and 1,000 weak-learners, both the MAS and the Laminating algorithms perform far better than the baselines. Performance tend to get similar for 10,000 stumps, which is unusually large.

As stated in § 5.2, the meta-parameters of the bandit methods have been optimized by running the training fully ten times, with the corresponding computational effort.

| Nb. of | Uniform | | | Bandits | | | MAS | | | |
|---|---|---|---|---|---|---|---|---|---|---|
| stumps | Naive | 1.Q | Q.1 | UCB⋆ | Exp3.P⋆ | ϵ-greedy⋆ | Naive | 1.Q | Q.1 | Laminating |
| MNIST | | | | | | | | | | |
| 10 | 51.18 (4.22) | 54.37 (7.93) | 48.15 (3.66) | 52.86 (4.75) | 53.80 (4.53) | 51.37 (6.35) | 25.91 (2.04) | 25.94 (2.57) | 25.73 (1.33) | 35.70 (2.35) |
| 100 | 8.95 (0.41) | 11.64 (1.06) | 8.69 (0.48) | 11.39 (0.53) | 11.58 (0.93) | 11.59 (1.12) | 4.87 (0.29) | 4.78 (0.16) | 4.54 (0.21) | 4.85 (0.16) |
| 1,000 | 1.75 (0.06) | 2.37 (0.12) | 1.76 (0.08) | 1.80 (0.08) | 2.18 (0.14) | 1.83 (0.16) | 1.50 (0.06) | 1.59 (0.08) | 1.45 (0.04) | 1.34 (0.08) |
| 10,000 | 0.94 (0.06) | 1.13 (0.03) | 0.94 (0.04) | 0.90 (0.05) | 0.84 (0.02) | 0.85 (0.07) | 0.92 (0.03) | 0.97 (0.05) | 0.94 (0.04) | 0.85 (0.04) |
| CIFAR-10 | | | | | | | | | | |
| 10 | 76.27 (0.97) | 78.57 (1.94) | 76.00 (1.60) | 77.04 (1.65) | 77.51 (1.50) | 77.13 (1.15) | 71.54 (0.69) | 71.13 (0.49) | 70.63 (0.34) | 71.54 (1.06) |
| 100 | 56.94 (1.01) | 58.33 (1.30) | 54.48 (0.64) | 57.49 (0.46) | 58.47 (0.81) | 58.19 (0.83) | 53.94 (0.55) | 52.79 (0.09) | 50.15 (0.64) | 50.44 (0.68) |
| 1,000 | 39.13 (0.61) | 39.97 (0.37) | 37.70 (0.38) | 38.13 (0.30) | 39.23 (0.31) | 38.36 (0.72) | 38.79 (0.28) | 38.31 (0.27) | 36.95 (0.25) | 36.39 (0.58) |
| 10,000 | 31.83 (0.29) | 31.16 (0.29) | 30.56 (0.30) | 30.55 (0.24) | 30.39 (0.22) | 29.96 (0.45) | 32.07 (0.27) | 31.36 (0.13) | 32.51 (0.38) | 31.17 (0.22) |

Table 3: Mean and standard deviation of the test error (in percent) on the two data-sets and for each method, estimated on ten randomized runs. Methods highlighted with a ⋆ require the tuning of meta-parameters which have been optimized by training fully multiple times.

On the MNIST data-set, when adding 10 or 100 weak-learners, our methods roughly divides the error rate by two, and still improves it by $\simeq 30\%$ with 1,000 stumps. The loss reduction follows the same pattern.

The CIFAR data-set is a very difficult pattern recognition problem. Still our algorithms performs substantially better than the baselines for 10 and 100 weak-learners, gaining more than $10\%$ in the test error rates, and behave similarly to the baselines for larger number of stumps.

As stated in § 1, the optimal values for a fixed product $QT$ moves to larger $T$ and smaller $Q$. For instance, On the MNIST data-set with MAS Naive, averaging on ten randomized runs, for respectively 10, 100, 1,000, 10,000 stumps, $T = 1,580, 13,030, 37,100, 43,600$, and $Q = 388, 73, 27, 19$. We obtain similar and consistent results across settings.

The overhead of MAS algorithms compared to Uniform ones is small, in our experiments, taking into account the time spent computing features, it is approximately $0.2\%$ for MAS Naive, $2\%$ for MAS 1.Q and $8\%$ for MAS Q.1. The Laminating algorithm has no overhead.

The poor behavior of bandit methods for small number of stumps may be related to the large variations of the sample weights during the first iterations of Boosting, which goes against the underlying assumption of stationarity of the loss reduction.

# 6   Conclusion

We have improved Boosting by modeling the statistical behavior of the weak-learners' edges. This allowed us to maximize the loss reduction under strict control of the computational cost. Experiments demonstrate that the algorithms perform well on real-world pattern recognition tasks.

Extensions of the proposed methods could be investigated along two axis. The first one is to blur the boundary between the MAS procedures and the Laminating, by deriving an analytical model of the loss reduction for generalized sampling procedures: Instead of doubling the number of samples and halving the number of weak-learners, we could adapt both set sizes optimally. The second is to add a bandit-like component to our methods by adding a variance term related to the lack of samples, and their obsolescence in the Boosting process. This would account for the degrading density estimation when weak-learner families have not been sampled for a while, and induce an exploratory sampling which may be missing in the current algorithms.

## Acknowledgments

This work was supported by the European Community's 7th Framework Programme under grant agreement 247022 – MASH, and by the Swiss National Science Foundation under grant 200021-124822 – VELASH. We also would like to thank Dr. Robert B. Israel, Associate Professor Emeritus at the University of British Columbia for his help on the derivation of the expectation of the true edge of the weak-learner with the highest approximated edge (equations (9) to (13)).

# References

[1] P. Auer, N. Cesa-Bianchi, and P. Fischer. Finite-time analysis of the multiarmed bandit problem. *Machine Learning*, 47(2):235–256, 2002.

[2] P. Auer, N. Cesa-Bianchi, Y. Freund, and R. Schapire. The nonstochastic multiarmed bandit problem. *SIAM Journal on Computing*, 32(1):48–77, 2003.

[3] R. Busa-Fekete and B. Kegl. Accelerating AdaBoost using UCB. *JMLR W&CP*, Jan 2009.

[4] R. Busa-Fekete and B. Kegl. Fast Boosting using adversarial bandits. In *ICML*, 2010.

[5] N. Duffield, C. Lund, and M. Thorup. Priority sampling for estimation of arbitrary subset sums. *J. ACM*, 54, December 2007.

[6] G. Escudero, L. Màrquez, and G. Rigau. Boosting applied to word sense disambiguation. *Machine Learning: ECML 2000*, pages 129–141, 2000.

[7] F. Fleuret and D. Geman. Stationary features and cat detection. *Journal of Machine Learning Research (JMLR)*, 9:2549–2578, 2008.

[8] Z. Kalal, J. Matas, and K. Mikolajczyk. Weighted sampling for large-scale Boosting. *British machine vision conference*, 2008.

[9] A. Krizhevsky. Learning Multiple Layers of Features from Tiny Images. Master's thesis, 2009. `http://www.cs.toronto.edu/~kriz/cifar.html`.

[10] Y. Lecun and C. Cortes. The mnist database of handwritten digits. `http://yann.lecun.com/exdb/mnist/`.

